# Radial Basis Functions: a Bayesian treatment

**David Barber\***         **Bernhard Schottky**

Neural Computing Research Group
Department of Applied Mathematics and Computer Science
Aston University, Birmingham B4 7ET, U.K.
http://www.ncrg.aston.ac.uk/
{D.Barber,B.Schottky}@aston.ac.uk

## Abstract

Bayesian methods have been successfully applied to regression and classification problems in multi-layer perceptrons. We present a novel application of Bayesian techniques to Radial Basis Function networks by developing a Gaussian approximation to the posterior distribution which, for fixed basis function widths, is analytic in the parameters. The setting of regularization constants by cross-validation is wasteful as only a single optimal parameter estimate is retained. We treat this issue by assigning prior distributions to these constants, which are then adapted in light of the data under a simple re-estimation formula.

## 1   Introduction

Radial Basis Function networks are popular regression and classification tools[10]. For fixed basis function centers, RBFs are linear in their parameters and can therefore be trained with simple one shot linear algebra techniques[10]. The use of unsupervised techniques to fix the basis function centers is, however, not generally optimal since setting the basis function centers using density estimation on the input data alone takes no account of the target values associated with that data. Ideally, therefore, we should include the target values in the training procedure[7, 3, 9]. Unfortunately, allowing centers to adapt to the training targets leads to the RBF being a nonlinear function of its parameters, and training becomes more problematic.

Most methods that perform supervised training of RBF parameters minimize the

training error, or penalized training error in the case of regularized networks[7, 3, 9]. The setting of the associated regularization constants is often achieved by computationally expensive approaches such as cross-validation which search through a set of regularization constants chosen a priori. Furthermore, much of the information contained in such computation is discarded in favour of keeping only a single regularization constant. A single set of RBF parameters is subsequently found by minimizing the penalized training error with the determined regularization constant. In this work, we assign prior distributions over these regularization constants, both for the hidden to output weights and the basis function centers. Together with a noise model, this defines an ideal Bayesian procedure in which the beliefs expressed in the distribution of regularization constants are combined with the information in the data to yield a posterior *distribution* of network parameters[6]. The beauty of this approach is that none of the information is discarded, in contrast to cross-validation type procedures. Bayesian techniques applied to such non-linear, non-parametric models, however, can also be computationally extremely expensive, as predictions require averaging over the high-dimensional posterior parameter distribution. One approach is to use Markov chain Monte Carlo techniques to draw samples from the posterior[8]. A simpler approach is the Laplace approximation which fits a Gaussian distribution with mean set to a mode of the posterior, and covariance set to the inverse Hessian evaluated at that mode. This can be viewed as a local posterior approximation, as the form of the posterior away from the mode does not affect the Gaussian fit. A third approach, called ensemble learning, also fits a Gaussian, but is based on a less local fit criterion, the Kullback-Leibler divergence[4, 5]. As shown in [1], this method can be applied successfully to multi-layer perceptrons, whereby the KL divergence is an *almost* analytic quantity in the adaptable parameters. For fixed basis function widths, the KL divergence for RBF networks is *completely* analytic in the adaptable parameters, leading to a relatively fast optimization procedure.

## 2 Bayesian Radial Basis Function Networks

For an $N$ dimensional input vector $\mathbf{x}$, we consider RBFs that compute the linear combination of $K$ Gaussian basis functions,

$$f(\mathbf{x}, \mathbf{m}) = \sum_{l=1}^{K} w_l \exp\left\{-\lambda_l \|\mathbf{x} - \mathbf{c}_l\|^2\right\} \tag{1}$$

where we denote collectively the centers $\mathbf{c}_1 \ldots \mathbf{c}_K$, and weights $\mathbf{w} = w_1 \ldots w_k$ by the parameter vector $\mathbf{m} = [\mathbf{c}_1', \ldots, \mathbf{c}_K', w_1, \ldots, w_K]'$. We consider the basis function widths $\lambda_1, \ldots \lambda_k$ to be fixed although, in principle, they can also be adapted by a similar technique to the one presented below. The data set that we wish to regress is a set of $P$ input-output pairs $D = \{\mathbf{x}^\mu, y^\mu, \mu = 1 \ldots P\}$. Assuming that the target outputs $y$ have been corrupted with additive Gaussian noise of variance $\beta^{-1}$, the likelihood of the data is[1]

$$p(D|\mathbf{m}, \beta) = \exp\left(-\beta E_D\right)/Z_D, \tag{2}$$

where the training error is defined,

$$E_D = \frac{1}{2} \sum_{\mu=1}^{P} \left(f(\mathbf{x}^\mu, \mathbf{m}) - y^\mu\right)^2 \tag{3}$$

To discourage overfitting, we choose a prior regularizing distribution for $\mathbf{m}$

$$p(\mathbf{m}|\alpha) = \exp\left(-E_m(\mathbf{m})\right)/Z_P \tag{4}$$

where we take $E_m(\mathbf{m}) = \frac{1}{2}\mathbf{m}^{\mathrm{T}}\mathrm{A}\mathbf{m}$ for a matrix A of hyperparameters. More complicated regularization terms, such as those that penalize centers that move away from specified points are easily incorporated in our formalism. For expositional clarity, we deal here with only the simple case of a diagonal regularizer matrix $\mathrm{A} = \alpha \mathrm{I}$.

The conditional distribution $p(\mathbf{m}|D, \alpha, \beta)$ is then given by

$$p(\mathbf{m}|D, \alpha, \beta) = \exp(-\beta E_D(\mathbf{m}) - E_m(\mathbf{m}))/Z_F \qquad (5)$$

We choose to model the hyperparameters $\alpha$ and $\beta$ by Gamma distributions,

$$p(\alpha) \propto \alpha^{a-1} e^{-\alpha/b} \qquad p(\beta) \propto \alpha^{c-1} e^{-\beta/d}, \qquad (6)$$

where $a, b, c, d$ are chosen constants. This completely specifies the joint posterior,

$$p(\mathbf{m}, \alpha, \beta|D) = p(\mathbf{m}|D, \alpha, \beta)p(\alpha)p(\beta). \qquad (7)$$

A Bayesian prediction for a new test point $\mathbf{x}$ is then given by the posterior average $\langle f(\mathbf{x}, \mathbf{m}) \rangle_{p(\mathbf{m}, \alpha, \beta|D)}$. If the centers are fixed, $p(\mathbf{w}|D, \alpha, \beta)$ is Gaussian and computing the posterior average is trivial. However, with adaptive centers, the posterior distribution is typically highly complex and computing this average is difficult[2]. We describe below approaches that approximate the posterior by a simpler distribution which can then be used to find the Bayesian predictions and error bars analytically.

## 3   Approximating the posterior

### 3.1   Laplace's method

Laplace's method is an approximation to the Bayesian procedure that fits a Gaussian to the mode $\mathbf{m}_0$ of $p(\mathbf{m}, |D, \alpha, \beta)$ by extremizing the exponent in (5)

$$T = \frac{\alpha}{2}||\mathbf{m}||^2 + \beta E_D(\mathbf{m}) \qquad (8)$$

with respect to $\mathbf{m}$. The mean of the approximating distribution is then set to the mode $\mathbf{m}_0$, and the covariance is taken to be the inverse Hessian around $\mathbf{m}_0$; this is then used to approximately compute the posterior average. This is a local method as no account is taken for the fit of the Gaussian away from the mode.

### 3.2   Kullback-Leibler method

The Kullback-Leibler divergence between the posterior $p(\mathbf{m}, \alpha, \beta|D)$ and an approximating distribution $q(\mathbf{m}, \alpha, \beta)$ is defined by

$$KL[q] = -\int q(\mathbf{m}, \alpha, \beta) \ln\left(\frac{p(\mathbf{m}, \alpha, \beta|D)}{q(\mathbf{m}, \alpha, \beta)}\right). \qquad (9)$$

$KL[q]$ is zero only if $p$ and $q$ are identical, and is greater than zero otherwise. Since in (5) $Z_F$ is unknown, we can compute the KL divergence only up to an additive constant, $L[q] = KL[q] - \ln Z_F$. We seek then a posterior approximation of the form $q(\mathbf{m}, \alpha, \beta) = Q(\mathbf{m})R(\alpha)S(\beta)$ where $Q(\mathbf{m})$ is Gaussian and the distributions $R$ and $S$ are determined by minimization of the functional $L[q]$[5].

We first consider optimizing $L$ with respect to the mean $\overline{\mathbf{m}}$ and covariance C of the Gaussian distribution $Q(\mathbf{m}) \propto \exp\left\{-\frac{1}{2}(\mathbf{m} - \overline{\mathbf{m}})^{\mathrm{T}}\mathrm{C}^{-1}(\mathbf{m} - \overline{\mathbf{m}})\right\}$. Omitting all constant terms and integrating out $\alpha$ and $\beta$, the $Q(\mathbf{m})$ dependency in $L$ is,

$$L[Q(\mathbf{m})] = -\int Q(\mathbf{m})\left[-\bar{\beta}E_D(\mathbf{m}) - \frac{1}{2}\bar{\alpha}||\mathbf{m}||^2 - \ln Q(\mathbf{m})\right]dm + \text{const}. \qquad (10)$$

where

$$\bar{\alpha} = \int \alpha R(\alpha)d\alpha, \qquad \bar{\beta} = \int \beta S(\beta)d\beta \qquad (11)$$

are the mean values of the hyperparameters. For Gaussian basis functions, the remaining integration in (10) over $Q(\mathbf{m})$ can be evaluated analytically, giving[3]

$$L[Q(\mathbf{m})] = \frac{1}{2}\bar{\alpha}\left\{\text{tr}(\mathbf{C}) + \|\bar{\mathbf{m}}\|^2\right\} + \bar{\beta}\langle E_D(\mathbf{m})\rangle_Q - \frac{1}{2}\ln(\det \mathbf{C}) + \text{const.} \qquad (12)$$

where

$$\langle E_D(\mathbf{m})\rangle_Q = \frac{1}{2}\sum_{\mu=1}^{P}\left((y^\mu)^2 - 2y^\mu\sum_{l=1}^{K}s_l^\mu + \sum_{kl=1}^{K}s_{kl}^\mu\right) \qquad (13)$$

The analytical formulae for

$$s_l^\mu = \langle w_l\exp\{-\lambda_l\|\mathbf{x}^\mu - \mathbf{c}_l\|^2\}\rangle_Q \qquad (14)$$

$$s_{kl}^\mu = \langle w_k w_l\exp\{-\lambda_k\|\mathbf{x}^\mu - \mathbf{c}_k\|^2\}\exp\{-\lambda_l\|\mathbf{x}^\mu - \mathbf{c}_l\|^2\}\rangle_Q \qquad (15)$$

are straightforward to compute, requiring only Gaussian integration[2]. The values for $\mathbf{C}$ and $\bar{\mathbf{m}}$ can then be found by optimizing (12).

We now turn to the functional optimisation of (9) with respect to $R$. Integrating out $\mathbf{m}$ and $\beta$ leaves, up to a constant,

$$L[R] = \int R(\alpha)\left\{\alpha\left[\frac{\|\bar{\mathbf{m}}\|^2}{2} + \frac{\text{tr}(\mathbf{C})}{2} + \frac{1}{b}\right] + \left[\frac{K(N+1)}{4} + a - 1\right]\ln\alpha + \ln R(\alpha)\right\}d\alpha \qquad (16)$$

As the first two terms in (16) constitute the log of a Gamma distribution (6), the functional (16) is optimized by choosing a Gamma distribution for $\alpha$,

$$R(\alpha) \propto \alpha^{r-1}e^{-\alpha/s} \qquad (17)$$

with

$$r = \frac{K(N+1)}{2} + a, \qquad \frac{1}{s} = \frac{\|\bar{\mathbf{m}}\|^2}{2} + \frac{1}{2}\text{tr}(\mathbf{C}) + \frac{1}{b}, \qquad \bar{\alpha} = rs. \qquad (18)$$

The same procedure for $S(\beta)$ yields

$$S(\beta) \propto \beta^{u-1}e^{-\beta/v} \qquad (19)$$

with

$$u = \frac{P}{2} + c, \qquad \frac{1}{v} = \langle E_D(\mathbf{m})\rangle_Q + \frac{1}{d}, \qquad \bar{\beta} = uv, \qquad (20)$$

where the averaged training error is given by (13). The optimization of the approximating distribution $Q(\mathbf{m})R(\alpha)S(\beta)$ can then be performed using an iterative procedure in which we first optimize (12) with respect to $\bar{\mathbf{m}}$ and $\mathbf{C}$ for fixed $\bar{\alpha}$, $\bar{\beta}$, and then update $\bar{\alpha}$ and $\bar{\beta}$ according to the re-estimation formulae (18,20).

After this iterative procedure has converged, we have an approximating distribution of parameters, both for the hidden to output weights and center positions (figure 1(a)). The actual predictions are then given by the posterior average over this distribution of networks. The model averaging effect inherent in the Bayesian procedure produces a final function potentially much more complex than that achievable by a single network.

A significant advantage of our procedure over the Laplace procedure is that we can lower bound model the likelihood $\ln p(D|\text{model}) \geq -(L + \ln Z_D + \ln Z_P)$. Hence, decreasing $L$ increases $p(D|\text{model})$. We can use this bound to rank different models, leading to principled Bayesian model selection.

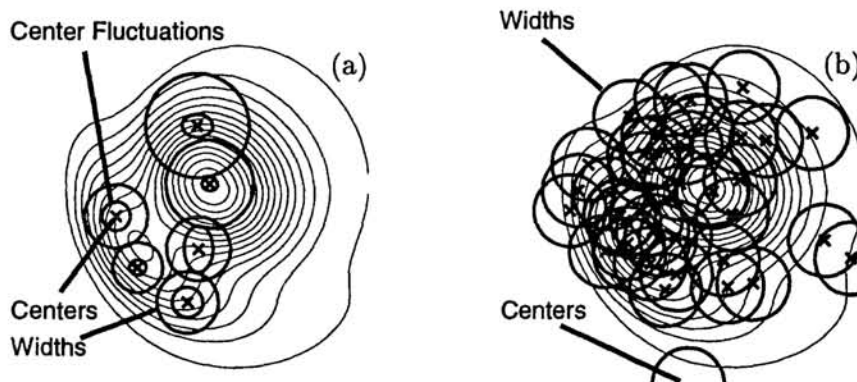

Figure 1: Regressing a surface from 40 noisy training examples. (a) The KL approximate Bayesian treatment fits 6 basis functions to the data. The posterior distribution for the parameters gives rise to a posterior weighted average of a distribution of the 6 Gaussians. We plot here the posterior standard deviation of the centers (center fluctuations) and the mean centers. The widths were fixed a priori using Maximum Likelihood. (b) Fixing a basis function on each training point with fixed widths. The hidden-output weights were determined by cross-validation of the penalised training error.

## 4 Relation to non-Bayesian treatments

One non-Bayesian approach to training RBFs is to minimze the training error (3) plus a regularizing term of the form (8) for fixed centers[7, 3, 9]. In figure 1(b) we fix a center on each training input. For fixed hyperparameters $\alpha$ and $\beta$, the optimal hidden-to-output weights can then be found by minimizing (8). To set the hyperparameters, we iterate this procedure using cross-validation. This results in a single estimate for the parameters $\mathbf{m}_0$ which is then used for predictions $f(\mathbf{x}, \mathbf{m}_0)$. In figure(1), both the Bayesian adaptive center and the fixed center methods have similar performance in terms of test error on this problem. However, the parsimonious representation of the data by the Bayesian adaptive center method may be advantageous if interpreting the data is important.

In principle, in the Bayesian approach, there is no need to carry out a cross-validation type procedure for the regularization parameters $\alpha, \beta$. After deciding on a particular Bayesian model with suitable hyperprior constants (here $a, b, c, d$), our procedure will combine these beliefs about the regularity of the RBF with the dataset in a principled manner, returning a-posteriori probabilities for the values of the regularization constants. Error bars on the predictions are easily calculated as the posterior distribution quantifies our uncertainty in the parameter estimates.

One way of viewing the connection between the CV and Bayesian approaches, is to identify the a-priori choice of CV regularization coefficients $\alpha_i$ that one wishes to examine as a uniform prior over the set $\{\alpha_i\}$. The posterior regularizer distribution is then a delta peak centred at that $\alpha_*$ with minimal CV error. This delta peak represents a loss of information regarding the performance of all the other networks trained with $\alpha_i \neq \alpha_*$. In contrast, in our Bayesian approach we assign a continuous prior distribution on $\alpha$, which is updated according to the evidence in the data. Any loss of information then occurs in approximating the resulting posterior distribution.

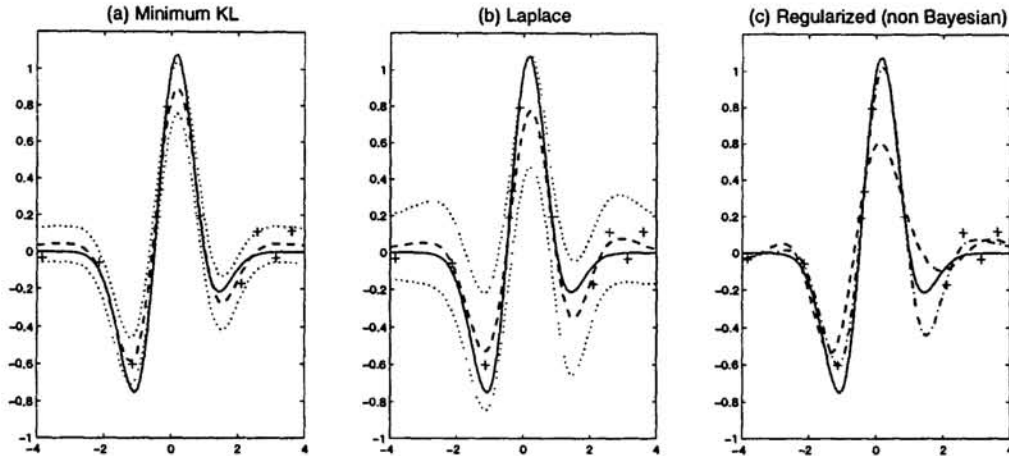

Figure 2: Minimal KL Gaussian fit, Laplace Gaussian, and a non-Bayesian procedure on regressing with 6 Gaussian basis functions. The training points are labelled by crosses and the target function $g$ is given by the solid lines. For both (a) and (b), the mean prediction is given by the dashed lines, and standard errors are given by the dots. (a) Approximate Bayesian solution based on Kullback-Leibler divergence. The regularization constant $\alpha$ and inverse noise level $\beta$ are adapted as described in the text. (b) Laplace method based on equation (8). Both $\alpha$ and $\beta$ are set to the mean of the hyperparameter distributions (6). The mean prediction is given by averaging over the locally approximated posterior. Note that the error bars are somewhat large, suggesting that the local posterior mass has been underestimated. (c) The broken line is the Laplace solution without averaging over the posterior, showing much greater variation than the averaged prediction in (b). The dashed line corresponds to fixing the basis function centers at each data point, and estimating the regularization constants $\alpha$ by cross-validation.

## 5   Demonstration

We apply the above outlined Bayesian framework to a simple one-dimensional regression problem. The function to be learned is given by

$$g(x) = (1 + x - 2x^2)\exp\{-x^2\}, \tag{21}$$

and is plotted in figure(2). The training patterns are sampled uniformly between $[-4, 4]$ and the output is corrupted with additive Gaussian noise of variance $\sigma^2 = 0.005$. The number of basis function is $K = 6$, giving a reasonably flexible model for this problem. In figure(2), we compare the Bayesian approaches (a),(b) to non-Bayesian approaches(c). In this demonstration, the basis function widths were chosen by penalised training error minimization and fixed throughout all experiments. For the Bayesian procedures, we chose hyperprior constants, $a = 2, b = 1/4, c = 4, d = 50$, corresponding to mean values $\bar{\alpha} = 0.5$ and $\bar{\beta} = 200$. In (c), we plot a more conventional approach using cross-validation to set the regularization constant.

A useful feature of the Bayesian approaches lies in the principled theory for the error bars. In (c), although we know the test error for each regularization constant in the set of constants we choose to examine, we do not know any principled procedure for using these values for error bar assessment.

# 6   Conclusions

We have incorporated Radial Basis Functions within a Bayesian framework, arguing that the selection of regularization constants by non-Bayesian methods such as cross-validation is wasteful of the information contained in our prior beliefs and the data set. Our framework encompasses flexible priors such as hard assigning a basis function center to each data point or penalizing centers that wander far from pre-assigned points. We have developed an approximation to the ideal Bayesian procedure by fitting a Gaussian distribution to the posterior based on minimizing the Kullback-Leibler divergence. This is an objectively better and more controlled approximation to the Bayesian procedure than the Laplace method. Furthermore, the KL divergence is an analytic quantity for fixed basis function widths. This framework also includes the automatic adaptation of regularization constants under the influence of data and provides a rigorous lower bound on the likelihood of the model.

## Acknowledgements

We would like to thank Chris Bishop and Chris Williams for useful discussions. BS thanks the Leverhulme Trust for support (F/250/K).

## Footnotes

\*Present address: SNN, University of Nijmegen, Geert Grooteplein 21, Nijmegen, The Netherlands. http://www.mbfys.kun.nl/snn/ email: davidb@mbfys.kun.nl

[1]In the following, $Z_D$, $Z_P$ and $Z_F$ are normalising constants

[2]The fixed and adaptive center Bayesian approaches are contrasted more fully in [2].

[3] $\langle\ldots\rangle_Q$ denotes $\int Q(\mathbf{m})\ldots d\mathbf{m}$

# References

[1] D. Barber and C. M. Bishop. On computing the KL divergence for Bayesian Neural Networks. Technical report, Neural Computing Research Group, Aston University, Birmingham, 1998. See also D. Barber and C. M. Bishop *These proceedings*.

[2] D. Barber and B. Schottky. Bayesian Radial Basis Functions. Technical report, Neural Computing Research Group, Aston University, Birmingham, 1998.

[3] C. M. Bishop. Improving the Generalization Properties of Radial Basis Function Networks. *Neural Computation*, 4(3):579–588, 1991.

[4] G. E. Hinton and D. van Camp. Keeping neural networks simple by minimizing the description length of the weights. In *Proceedings of the Seventh Annual ACM Workshop on Computational Learning Theory (COLT '93)*, 1993.

[5] D. J. C. MacKay. Developments in probabilistic modelling with neural networks – ensemble learning. In *Neural Networks: Artificial Intelligence and Industrial Applications. Proceedings of the 3rd Annual Symposium on Neural Networks, Nijmegan, Netherlands, 14-15 September 1995*, pages 191–198. Springer.

[6] D. J. C. MacKay. Bayesian Interpolation. *Neural Computation*, 4(3):415–447, 1992.

[7] J. Moody and C. J. Darken. Fast Learning in Networks of Locally-Tuned Processing Units. *Neural Computation*, 1:281–294, 1989.

[8] Neal, R. M. *Bayesian Learning for Neural Networks*. Springer, New York, 1996. Lecture Notes in Statistics 118.

[9] M. J. L. Orr. Regularization in the Selection of Radial Basis Function Centers. *Neural Computation*, 7(3):606–623, 1995.

[10] M. J. L. Orr. Introduction to Radial Basis Function Networks. Technical report, Centre for Cognitive Science, Univeristy of Edinburgh, Edinburgh, EH8 9LW, U.K., 1996.